# Memory-based Stochastic Optimization

**Andrew W. Moore and Jeff Schneider**
School of Computer Science
Carnegie-Mellon University
Pittsburgh, PA 15213

### Abstract

In this paper we introduce new algorithms for optimizing noisy plants in which each experiment is very expensive. The algorithms build a global non-linear model of the expected output at the same time as using Bayesian linear regression analysis of locally weighted polynomial models. The local model answers queries about confidence, noise, gradient and Hessians, and use them to make automated decisions similar to those made by a practitioner of Response Surface Methodology. The global and local models are combined naturally as a locally weighted regression. We examine the question of whether the global model can really help optimization, and we extend it to the case of time-varying functions. We compare the new algorithms with a highly tuned higher-order stochastic optimization algorithm on randomly-generated functions and a simulated manufacturing task. We note significant improvements in total regret, time to converge, and final solution quality.

## 1 INTRODUCTION

In a stochastic optimization problem, noisy samples are taken from a plant. A sample consists of a chosen control $\mathbf{u}$ (a vector of real numbers) and a noisy observed response $y$. $y$ is drawn from a distribution with mean and variance that depend on $\mathbf{u}$. $y$ is assumed to be independent of previous experiments. Informally the goal is to quickly find control $\mathbf{u}$ to maximize the expected output $E[y \mid \mathbf{u}]$. This is different from conventional numerical optimization because the samples can be very noisy, there is no gradient information, and we usually wish to avoid ever performing badly (relative to our start state) even during optimization. Finally and importantly: **each experiment is very expensive** and there is **ample computational time** (often many minutes) for deciding on the next experiment. The following questions are both interesting and important: how should this computational time best be used, and how can the data best be used?

Stochastic optimization is of real industrial importance, and indeed one of our reasons for investigating it is an association with a U.S. manufacturing company

that has many new examples of stochastic optimization problems every year.

The discrete version of this problem, in which **u** is chosen from a discrete set, is the well known $k$-armed bandit problem. Reinforcement learning researchers have recently applied bandit-like algorithms to efficiently optimize several discrete problems [Kaelbling, 1990, Greiner and Jurisica, 1992, Gratch *et al.*, 1993, Maron and Moore, 1993]. This paper considers extensions to the continuous case in which **u** is a vector of reals. We anticipate useful applications here too. Continuity implies a formidable number of arms (uncountably infinite) but permits us to assume smoothness of $E[y \mid \mathbf{u}]$ as a function of **u**.

The most popular current techniques are:

- **Response Surface Methods (RSM).** Current RSM practice is described in the classic reference [Box and Draper, 1987]. Optimization proceeds by cautious steepest ascent hill-climbing. A region of interest (ROI) is established at a starting point and experiments are made at positions within the region that can best be used to identify the function properties with low-order polynomial regression. A large portion of the RSM literature concerns *experimental design*—the decision of where to take data points in order to acquire the lowest variance estimate of the local polynomial coefficients in a fixed number of experiments. When the gradient is estimated with sufficient confidence, the ROI is moved accordingly. Regression of a quadratic locates optima within the ROI and also diagnoses ridge systems and saddle points.

  The strength of RSM is that it is careful not to change operating conditions based on inadequate evidence, but moves once the data justifies. A weakness of RSM is that human judgment is needed: it is not an algorithm, but a manufacturing methodology.

- **Stochastic Approximation methods.** The algorithm of [Robbins and Monro, 1951] does root finding without the use of derivative estimates. Through the use of successively smaller steps convergence is proven under broad assumptions about noise. Keifer-Wolfowitz (KW) [Kushner and Clark, 1978] is a related algorithm for optimization problems. From an initial point it estimates the gradient by performing an experiment in each direction along each dimension of the input space. Based on the estimate, it moves its experiment center and repeats. Again, use of decreasing step sizes leads to a proof of convergence to a local optimum.

  The strength of KW is its aggressive exploration, its simplicity, and that it comes with convergence guarantees. However, it has more of a danger of attempting wild experiments in the presence of noise, and effectively discards the data it collects after each gradient estimate is made. In practice, higher order versions of KW are available in which convergence is accelerated by replacing the fixed step size schedule with an adaptive one [Kushner and Clark, 1978]. Later we compare the performance of our algorithms to such a higher-order KW.

## 2   MEMORY-BASED OPTIMIZATION

Neither KW nor RSM uses old data. After a gradient has been identified the control **u** is moved up the gradient and the data that produced the gradient estimate is discarded. Does this lead to inefficiencies in operation? This paper investigates one way of using old data: build a global non-linear plant model with it.

We use locally weighted regression to model the system [Cleveland and Delvin, 1988, Atkeson, 1989, Moore, 1992]. We have adapted the methods to return posterior distributions for their coefficients and noise (and thus, indirectly, their predictions)

based on very broad priors, following the Bayesian methods for global linear regression described in [DeGroot, 1970].

We estimate the coefficients $\beta = \{\beta_1 \ldots \beta_m\}$ of a local polynomial model in which the data was generated by the polynomial and corrupted with gaussian noise of variance $\sigma^2$, which we also estimate. Our prior assumption will be that $\beta$ is distributed according to a multivariate gaussian of mean $\mathbf{0}$ and covariance matrix $\mathbf{\Sigma}$. Our prior on $\sigma$ is that $1/\sigma^2$ has a gamma distribution with parameters $\alpha$ and $\beta$.

Assume we have observed $n$ pieces of data. The $j$th polynomial term for the $i$th data point is $X_{ij}$ and the output response of the $i$th data point is $Y_i$. Assume further that we wish to estimate the model local to the query point $\mathbf{x}_q$, in which a data point at distance $d_i$ from the the query point has weight $w_i = \exp(-d_i^2/K)$. $K$, the *kernel width* is a fixed parameter that determines the degree of localness in the local regression. Let $\mathbf{W} = \mathrm{Diag}(w_1, w_2 \ldots w_n)$.

The marginal posterior distribution of $\beta$ is a $t$ distribution with mean $\bar{\beta} = (\mathbf{\Sigma}^{-1} + X^T W^2 X)^{-1}(X^T W^2 Y)$ covariance

$$(2\beta + (Y^T - \beta^T X^T)W^2 Y^T)(\mathbf{\Sigma}^{-1} + X^T W^2 X)^{-1} / (2\alpha + \sum_{i=1}^{n} w_i^2) \qquad (1)$$

and $\alpha + \sum_{i=1}^{n} w_i^2$ degrees of freedom.

We assume a wide, weak, prior $\mathbf{\Sigma} = \mathrm{Diag}(20^2, 20^2, \ldots 20^2)$, $\alpha = 0.8$, $\beta = 0.001$, meaning the prior assumes each regression coefficient independently lies with high probability in the range -20 to 20, and the noise lies in the range 0.01 to 0.5.

Briefly, we note the following reasons that Bayesian locally weighted polynomial regression is particularly suited to this application:

- We can directly obtain meaningful confidence estimates of the joint pdf of the regressed coefficients and predictions. Indirectly, we can compute the probability distribution of the steepest gradient, the location of local optima and the principal components of the local Hessian.

- The Bayesian approach allows meaningful regressions even with fewer data points than regression coefficients—the posterior distribution reveals enormous lack of confidence in some aspects of such a model but other useful aspects can still be predicted with confidence. This is crucial in high dimensions, where it may be more effective to head in a known positive gradient without waiting for all the experiments that would be needed for a precise estimate of steepest gradient.

- Other pros and cons of locally weighted regression in the context of control can be found in [Moore *et al.*, 1995].

Given the ability to derive a plant model from data, how should it best be used? The true optimal answer, which requires solving an infinite-dimensional Markov decision process, is intractable. We have developed four approximate algorithms that use the learned model, described briefly below.

- **AutoRSM.** Fully automates the (normally manual) RSM procedure and incorporates weighted data from the model; not only from the current design. It uses online experimental design to pick ROI design points to maximize information about local gradients and optima. Space does not permit description of the linear algebraic formulations of these questions.

- **PMAX.** This is a greedy, simpler approach that uses the global non-linear model from the data to jump immediately to the model optimum. This is similar to the technique described in [Botros, 1994], with two extensions. First, the Bayesian

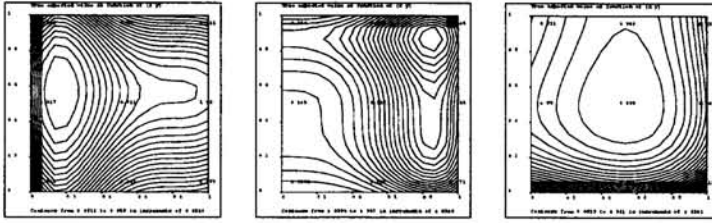

Figure 1: Three examples of 2-d functions used in optimization experiments

priors enable useful decisions before the regression becomes full-rank. Second, local quadratic models permit second-order convergence near an optimum.

- **IEMAX.** Applies Kaelbling's IE algorithm [Kaelbling, 1990] in the continuous case using Bayesian confidence intervals.

$$\mathbf{u}_{\text{chosen}} = \underset{\mathbf{u}}{\text{argmax}} \ \hat{f}_{\text{opt}}(\mathbf{u}) \tag{2}$$

where $\hat{f}_{\text{opt}}(\mathbf{u})$ is the top of the 95th %-ile confidence interval. The intuition here is that we are encouraged to explore more aggressively than PMAX, but will not explore areas that are confidently below the best known optimum.

- **COMAX.** In a real plant we would never want to apply PMAX or IEMAX. Experiments must be cautious for reasons of safety, quality control, and managerial peace of mind. COMAX extends IEMAX thus:

$$\mathbf{u}_{\text{chosen}} = \underset{\mathbf{u} \in SAFE}{\text{argmax}} \ \hat{f}_{\text{opt}}(\mathbf{u}); \mathbf{u} \in SAFE \Leftrightarrow \hat{f}_{\text{pess}}(\mathbf{u}) > \text{disaster threshold} \tag{3}$$

Analysis of these algorithms is problematic unless we are prepared to make strong assumptions about the form of $E[Y \mid \mathbf{u}]$. To examine the general case we rely on Monte Carlo simulations, which we now describe.

The experiments used randomly generated nonlinear unimodal (but not necessarily convex) $d$-dimensional functions from $[0,1]^d \to [0,1]$. Figure 1 shows three example 2-d functions. Gaussian noise ($\sigma = 0.1$) is added to the functions. This is large noise, and means several function evaluations would be needed to achieve a reliable gradient estimate for a system using even a large step size such as 0.2.

The following optimization algorithms were tested on a sample of such functions.

| | |
|---|---|
| **Vary-KW** | The best performing KW algorithm we could find varied step size and adapted gradient estimation steps to avoid undue regret at optima. |
| **Fixed-KW** | A version of KW that keeps its gradient-detecting step size fixed. This risks causing extra regret at a true optima, but has less chance of becoming delayed by a non-optimum. |
| **Auto-RSM** | The best performing version thereof. |
| **Passive-RSM** | Auto-RSM continues to identify the precise location of the optimum when it's arrived at that optimum. When Passive-RSM is confident (greater than 99%) that it knows the location of the optimum to two significant places, it stops experimenting. |
| **Linear RSM** | A linear instead of quadratic model, thus restricted to steepest ascent. |
| **CRSM** | Auto-RSM with conservative parameters, more typical of those recommended in the RSM literature. |
| **Pmax, IEmax and Comax** | As described above. |

Figures 2a and 2b show the first sixty experiments taken by AutoRSM and KW respectively on their journeys to the goal.

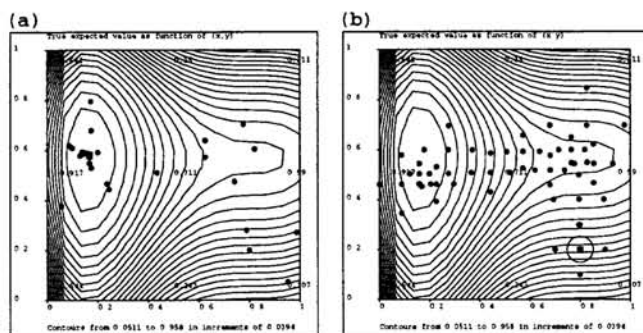

Figure 2a: The path taken (start at (0.8,0.2)) by AutoRSM optimizing the given function with added noise of standard deviation 0.1 at each experiment.

Figure 2b: The path taken (start at (0.8,0.2)) by KW. KW's path looks deceptively bad, but remember it is continually buffeted by considerable noise.

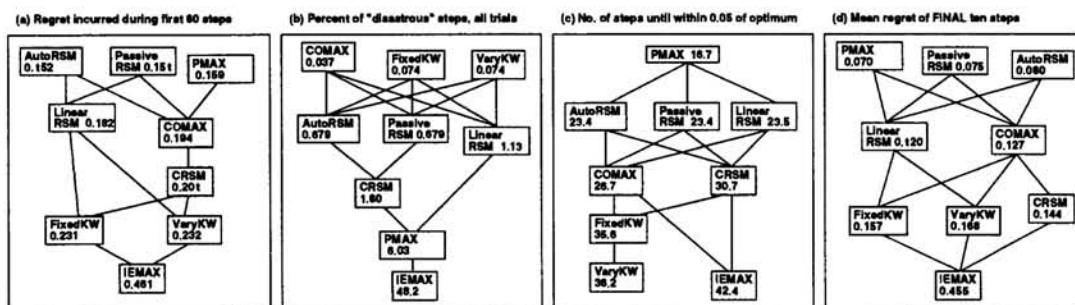

Figure 3: Comparing nine stochastic optimization algorithms by four criteria: (a) Regret, (b) Disasters, (c) Speed to converge (d) Quality at convergence. The partial order depicted shows which results are significant at the 99% level (using blocked pairwise comparisons). The outputs of the random functions range between 0–1 over the input domain. The numbers in the boxes are means over fifty 5-d functions. (a) Regret is defined as the mean $y_{opt} - y_i$—the cost incurred during the optimization compared with performance if we had known the optimum location and used it from the beginning. With the exception of IEMAX, model-based methods perform significantly better than KW, with reduced advantage for cautious and linear methods. (b) The %-age of steps which tried experiments with more than 0.1 units worse performance than at the search start. This matters to a risk averse manager. AutoRSM has fewer than 1% disasters, but COMAX and the model-free methods do better still. PMAX's aggressive exploration costs it. (c) The number of steps until we reach within 0.05 units of optimal. PMAX's aggressiveness wins. (d) The quality of the "final" solution between steps 50 and 60 of the optimization.

Results for 50 trials of each optimization algorithms for five-dimensional randomly generated functions are depicted in Figure 3. Many other experiments were performed in other dimensionalities and for modified versions of the algorithm, but space does not permit detailed discussion here.

Finally we performed experiments with the simulated power-plant process in Figure 4. The catalyst controller adjusts the flow rate of the catalyst to achieve the goal chemical A content. Its actions also affect chemical B content. The temperature controller adjusts the reaction chamber temperature to achieve the goal chemical B content. The chemical contents are also affected by the flow rate which is determined externally by demand for the product.

The task is to find the optimal values for the six controller parameters that minimize the total squared deviation from desired values of chemical A and chemical B contents. The feedback loops from sensors to controllers have significant delay. The controller gains on product demand are feedforward terms since there is significant delay in the effects of demand on the process. Finally, the performance of the system may also depend on variations over time in the composition of the input chemicals which can not be directly sensed.

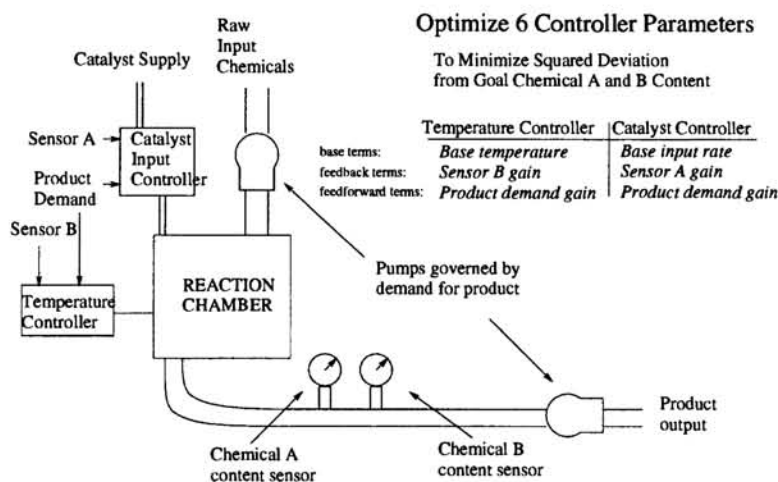

Figure 4: A Simulated Chemical Process

The total summed regrets of the optimization methods on 200 simulated steps were:

| StayAtStart | FixedKW | AutoRSM | PMAX | COMAX |
|---|---|---|---|---|
| 10.86 | 2.82 | 1.32 | 3.30 | 4.50 |

In this case AutoRSM is best, considerably beating the best KW algorithm we could find. In contrast PMAX and COMAX did poorly: in this plant wild experiments are very costly to PMAX and COMAX is too cautious. StayAtStart is the regret that would be incurred if all 200 steps were taken at the initial parameter setting.

## 3   UNOBSERVED DISTURBANCES

An apparent danger of learning a model is that if the environment changes, the out of date model will mean poor performance and very slow adaptation. The model-free methods, which use only recent data, will react more nimbly. A simple but unsatisfactory answer to this is to use a model that implicitly (e.g. a neural net) or explicitly (e.g. local weighted regression of the fifty most recent points) forgets. An interesting possibility is to learn a model in a way that automatically determines whether a disturbance has occurred, and if so, how far back to forget.

The following "adaptive forgetting" (AF) algorithm was added to the AutoRSM algorithm: At each step, use all the previous data to generate 99% confidence intervals on the output value at the current step. If the observed output is outside the intervals assume that a large change in the system has occured and forget all previous data. This algorithm is good for recognizing jumps in the plant's operating characteristics and allows AutoRSM to respond to them quickly, but is not suitable for detecting and handling process drift.

We tested our algorithm's performance on the simulated plant for 450 steps. Operation began as before, but at step 150 there was an unobserved change in the composition of the raw input chemicals. The total regrets of the optimization methods were:

| StayAtStart | FixedKW | AutoRSM | PMAX | AutoRSM/AF |
|---|---|---|---|---|
| 11.90 | 5.31 | 8.37 | 9.23 | 2.75 |

AutoRSM and PMAX do poorly because all their decisions after step 150 are based partially on the invalid data collected before then. The AF addition to AutoRSM solves the problem while beating the best KW by a factor of 2. Furthermore, AutoRSM/AF gets 1.76 on the invariant task, thus demonstrating that it can be used safely in cases where it is not known if the process is time varying.

## 4  DISCUSSION

Botros' thesis [Botros, 1994] discusses an algorithm similar to PMAX based on local linear regression. [Salganicoff and Ungar, 1995] uses a decision tree to learn a model. They use Gittins indices to suggest experiments: we believe that the memory-based methods can benefit from them too. They, however, do not use gradient information, and so require many experiments to search a 2D space.

IEmax performed badly in these experiments, but optimism-guided exploration may prove important in algorithms which check for potentially superior local optima.

A possible extension is self tuning optimization. Part way through an optimization, to estimate the best optimization parameters for an algorithm we can run monte-carlo simulations which run on sample functions from the posterior global model given the current data.

This paper has examined the question of how much can learning a Bayesian memory-based model accelerate the convergence of stochastic optimization. We have proposed four algorithms for doing this, one based on an autonomous version of RSM; the other three upon greedily jumping to optima of three criteria dependent on predicted output and uncertainty. Empirically the model-based methods provide significant gains over a highly tuned higher order model-free method.

## References

[Atkeson, 1989] C. G. Atkeson. Using Local Models to Control Movement. In *Proceedings of Neural Information Processing Systems Conference*, November 1989.

[Botros, 1994] S. M. Botros. Model-Based Techniques in Motor Learning and Task Optimization. PhD. Thesis, MIT Dept. of Brain and Cognitive Sciences, February 1994.

[Box and Draper, 1987] G. E. P. Box and N. R. Draper. *Empirical Model-Building and Response Surfaces*. Wiley, 1987.

[Cleveland and Delvin, 1988] W. S. Cleveland and S. J. Delvin. Locally Weighted Regression: An Approach to Regression Analysis by Local Fitting. *Journal of the American Statistical Association*, 83(403):596–610, September 1988.

[DeGroot, 1970] M. H. DeGroot. *Optimal Statistical Decisions*. McGraw-Hill, 1970.

[Gratch et al., 1993] J. Gratch, S. Chien, and G. DeJong. Learning Search Control Knowledge for Deep Space Network Scheduling. In *Proceedings of the 10th International Conference on Machine Learning*. Morgan Kaufmann, June 1993.

[Greiner and Jurisica, 1992] R. Greiner and I. Jurisica. A statistical approach to solving the EBL utility problem. In *Proceedings of the Tenth International Conference on Artificial Intelligence (AAAI-92)*. MIT Press, 1992.

[Kaelbling, 1990] L. P. Kaelbling. Learning in Embedded Systems. PhD. Thesis; Technical Report No. TR-90-04, Stanford University, Department of Computer Science, June 1990.

[Kushner and Clark, 1978] H. Kushner and D. Clark. *Stochastic Approximation Methods for Constrained and Unconstrained Systems*. Springer-Verlag, 1978.

[Maron and Moore, 1993] O. Maron and A. Moore. Hoeffding Races: Accelerating Model Selection Search for Classification and Function Approximation. In *Advances in Neural Information Processing Systems 6*. Morgan Kaufmann, December 1993.

[Moore et al., 1995] A. W. Moore, C. G. Atkeson, and S. Schaal. Memory-based Learning for Control. Technical report, CMU Robotics Institute, Technical Report CMU-RI-TR-95-18 *(Submitted for Publication)*, 1995.

[Moore, 1992] A. W. Moore. Fast, Robust Adaptive Control by Learning only Forward Models. In J. E. Moody, S. J. Hanson, and R. P. Lippman, editors, *Advances in Neural Information Processing Systems 4*. Morgan Kaufmann, April 1992.

[Robbins and Monro, 1951] H. Robbins and S. Monro. A stochastic approximation method. *Annals of Mathematical Statistics*, 22:400–407, 1951.

[Salganicoff and Ungar, 1995] M. Salganicoff and L. H. Ungar. Active Exploration and Learning in Real-Valued Spaces using Multi-Armed Bandit Allocation Indices. In *Proceedings of the 12th International Conference on Machine Learning*. Morgan Kaufmann, 1995.
